# Discriminative K-means for Clustering

**Jieping Ye**
Arizona State University
Tempe, AZ 85287
jieping.ye@asu.edu

**Zheng Zhao**
Arizona State University
Tempe, AZ 85287
zhaozheng@asu.edu

**Mingrui Wu**
MPI for Biological Cybernetics
Tübingen, Germany
mingrui.wu@tuebingen.mpg.de

## Abstract

We present a theoretical study on the discriminative clustering framework, recently proposed for simultaneous subspace selection via linear discriminant analysis (LDA) and clustering. Empirical results have shown its favorable performance in comparison with several other popular clustering algorithms. However, the inherent relationship between subspace selection and clustering in this framework is not well understood, due to the iterative nature of the algorithm. We show in this paper that this iterative subspace selection and clustering is equivalent to kernel K-means with a specific kernel Gram matrix. This provides significant and new insights into the nature of this subspace selection procedure. Based on this equivalence relationship, we propose the Discriminative K-means (DisKmeans) algorithm for simultaneous LDA subspace selection and clustering, as well as an automatic parameter estimation procedure. We also present the nonlinear extension of DisKmeans using kernels. We show that the learning of the kernel matrix over a convex set of pre-specified kernel matrices can be incorporated into the clustering formulation. The connection between DisKmeans and several other clustering algorithms is also analyzed. The presented theories and algorithms are evaluated through experiments on a collection of benchmark data sets.

## 1   Introduction

Applications in various domains such as text/web mining and bioinformatics often lead to very high-dimensional data. Clustering such high-dimensional data sets is a contemporary challenge, due to the curse of dimensionality. A common practice is to project the data onto a low-dimensional subspace through unsupervised dimensionality reduction such as Principal Component Analysis (PCA) [9] and various manifold learning algorithms [1, 13] before the clustering. However, the projection may not necessarily improve the separability of the data for clustering, due to the inherent separation between subspace selection (via dimensionality reduction) and clustering.

One natural way to overcome this limitation is to integrate dimensionality reduction and clustering in a joint framework. Several recent work [5, 10, 16] incorporate supervised dimensionality reduction such as Linear Discriminant Analysis (LDA) [7] into the clustering framework, which performs clustering and LDA dimensionality reduction simultaneously. The algorithm, called Discriminative Clustering (DisCluster) in the following discussion, works in an iterative fashion, alternating between LDA subspace selection and clustering. In this framework, clustering generates the class labels for LDA, while LDA provides the subspace for clustering. Empirical results have shown the benefits of clustering in a low dimensional discriminative space rather than in the principal component space (generative). However, the integration between subspace selection and clustering in DisCluster is not well understood, due to the intertwined and iterative nature of the algorithm.

In this paper, we analyze this discriminative clustering framework by studying several fundamental and important issues: (1) What do we really gain by performing clustering in a low dimensional discriminative space? (2) What is the nature of its iterative process alternating between subspace

selection and clustering? (3) Can this iterative process be simplified and improved? (4) How to estimate the parameter involved in the algorithm?

The main contributions of this paper are summarized as follows: (1) We show that the LDA projection can be factored out from the integrated LDA subspace selection and clustering formulation. This results in a simple trace maximization problem associated with a regularized Gram matrix of the data, which is controlled by a regularization parameter $\lambda$; (2) The solution to this trace maximization problem leads to the Discriminative K-means (DisKmeans) algorithm for simultaneous LDA subspace selection and clustering. DisKmeans is shown to be equivalent to kernel K-means, where discriminative subspace selection essentially constructs a kernel Gram matrix for clustering. This provides new insights into the nature of this subspace selection procedure; (3) The DisKmeans algorithm is dependent on the value of the regularization parameter $\lambda$. We propose an automatic parameter tuning process (model selection) for the estimation of $\lambda$; (4) We propose the nonlinear extension of DisKmeans using the kernels. We show that the learning of the kernel matrix over a convex set of pre-specified kernel matrices can be incorporated into the clustering formulation, resulting in a semidefinite programming (SDP) [15]. We evaluate the presented theories and algorithms through experiments on a collection of benchmark data sets.

## 2 Linear Discriminant Analysis and Discriminative Clustering

Consider a data set consisting of $n$ data points $\{x_i\}_{i=1}^n \in \mathbb{R}^m$. For simplicity, we assume the data is centered, that is, $\sum_{i=1}^n x_i/n = 0$. Denote $X = [x_1, \cdots, x_n]$ as the data matrix whose $i$-th column is given by $x_i$. In clustering, we aim to group the data $\{x_i\}_{i=1}^n$ into $k$ clusters $\{C_j\}_{j=1}^k$. Let $F \in \mathbb{R}^{n \times k}$ be the cluster indicator matrix defined as follows:

$$F = \{f_{i,j}\}_{n \times k}, \text{ where } f_{i,j} = 1, \text{ iff } x_i \in C_j. \tag{1}$$

We can define the weighted cluster indicator matrix as follows [4]:

$$L = [L_1, L_2, \cdots, L_k] = F(F^T F)^{-\frac{1}{2}}. \tag{2}$$

It follows that the $j$-th column of $L$ is given by

$$L_j = (0, \ldots, 0, \overbrace{1, \ldots, 1}^{n_j}, 0, \ldots, 0)^T / n_j^{\frac{1}{2}}, \tag{3}$$

where $n_j$ is the sample size of the $j$-th cluster $C_j$. Denote $\mu_j = \sum_{x \in C_j} x/n_j$ as the mean of the $j$-th cluster $C_j$. The within-cluster scatter, between-cluster scatter, and total scatter matrices are defined as follows [7]:

$$S_w = \sum_{j=1}^k \sum_{x_i \in C_j} (x_i - \mu_j)(x_i - \mu_j)^T, \quad S_b = \sum_{j=1}^k n_j \mu_j \mu_j^T = XLL^T X^T, \quad S_t = XX^T. \tag{4}$$

It follows that trace($S_w$) captures the intra-cluster distance, and trace($S_b$) captures the inter-cluster distance. It can be shown that $S_t = S_w + S_b$.

Given the cluster indicator matrix $F$ (or $L$), Linear Discriminant Analysis (LDA) aims to compute a linear transformation (projection) $P \in \mathbb{R}^{m \times d}$ that maps each $x_i$ in the $m$-dimensional space to a vector $\hat{x}_i$ in the $d$-dimensional space ($d < m$) as follows: $x_i \in \mathbb{R}^m \rightarrow \hat{x}_i = P^T x_i \in \mathbb{R}^d$, such that the following objective function is maximized [7]: trace $\left((P^T S_w P)^{-1} P^T S_b P\right)$. Since $S_t = S_w + S_b$, the optimal transformation matrix $P$ is also given by maximizing the following objective function:

$$\text{trace} \left((P^T S_t P)^{-1} P^T S_b P\right). \tag{5}$$

For high-dimensional data, the estimation of the total scatter (covariance) matrix is often not reliable. The regularization technique [6] is commonly applied to improve the estimation as follows:

$$\tilde{S}_t = S_t + \lambda I_m = XX^T + \lambda I_m, \tag{6}$$

where $I_m$ is the identity matrix of size $m$ and $\lambda > 0$ is a regularization parameter.

In Discriminant Clustering (DisCluster) [5, 10, 16], the transformation matrix $P$ and the weighted cluster indicator matrix $L$ are computed by maximizing the following objective function:

$$\begin{aligned} f(L, P) &\equiv \text{trace} \left((P^T \tilde{S}_t P)^{-1} P^T S_b P\right) \\ &= \text{trace} \left((P^T (XX^T + \lambda I_m)P)^{-1} P^T XLL^T X^T P\right). \end{aligned} \tag{7}$$

The algorithm works in an intertwined and iterative fashion, alternating between the computation of $L$ for a given $P$ and the computation of $P$ for a given $L$. More specifically, for a given $L$, $P$ is given by the standard LDA procedure. Since trace$(AB)$ = trace$(BA)$ for any two matrices [8], for a given $P$, the objective function $f(L, P)$ can be expressed as:

$$f(L, P) = \text{trace} \left( L^T X^T P (P^T (XX^T + \lambda I_m) P)^{-1} P^T X L \right). \tag{8}$$

Note that $L$ is not an arbitrary matrix, but a weighted cluster indicator matrix, as defined in Eq. (3). The optimal $L$ can be computed by applying the gradient descent strategy [10] or by solving a kernel K-means problem [5, 16] with $X^T P (P^T (XX^T + \lambda I_m) P)^{-1} P^T X$ as the kernel Gram matrix [4]. The algorithm is guaranteed to converge in terms of the value of the objective function $f(L, P)$, as the value of $f(L, P)$ monotonically increases and is bounded from above.

Experiments [5, 10, 16] have shown the effectiveness of DisCluster in comparison with several other popular clustering algorithms. However, the inherent relationship between subspace selection via LDA and clustering is not well understood, and there is need for further investigation. We show in the next section that the iterative subspace selection and clustering in DisCluster is equivalent to kernel K-means with a specific kernel Gram matrix. Based on this equivalence relationship, we propose the Discriminative K-means (DisKmeans) algorithm for simultaneous LDA subspace selection and clustering.

## 3 DisKmeans: Discriminative K-means with a Fixed $\lambda$

Assume that $\lambda$ is a fixed positive constant. Let's consider the maximization of the function in Eq. (7):

$$f(L, P) = \text{trace} \left( (P^T (XX^T + \lambda I_m) P)^{-1} P^T XLL^T X^T P \right). \tag{9}$$

Here, $P$ is a transformation matrix and $L$ is a weighted cluster indicator matrix as in Eq. (3). It follows from the Representer Theorem [14] that the optimal transformation matrix $P \in \mathbb{R}^{m \times d}$ can be expressed as $P = XH$, for some matrix $H \in \mathbb{R}^{n \times d}$. Denote $G = X^T X$ as the Gram matrix, which is symmetric and positive semidefinite. It follows that

$$f(L, P) = \text{trace} \left( \left( H^T (GG + \lambda G) H \right)^{-1} H^T GLL^T GH \right). \tag{10}$$

We show that the matrix $H$ can be factored out from the objective function in Eq. (10), thus dramatically simplifying the optimization problem in the original DisCluster algorithm. The main result is summarized in the following theorem:

**Theorem 3.1.** *Let $G$ be the Gram matrix defined as above and $\lambda > 0$ be the regularization parameter. Let $L^*$ and $P^*$ be the optimal solution to the maximization of the objective function $f(L, P)$ in Eq. (7). Then $L^*$ solves the following maximization problem:*

$$L^* = \arg \max_L \ trace \left( L^T \left( I_n - (I_n + \frac{1}{\lambda} G)^{-1} \right) L \right). \tag{11}$$

*Proof.* Let $G = U \Sigma U^T$ be the Singular Value Decomposition (SVD) [8] of $G$, where $U \in \mathbb{R}^{n \times n}$ is orthogonal, $\Sigma = \text{diag} (\sigma_1, \cdots, \sigma_t, 0, \cdots, 0) \in \mathbb{R}^{n \times n}$ is diagonal, and $t = \text{rank}(G)$. Let $U_1 \in \mathbb{R}^{n \times t}$ consist of the first $t$ columns of $U$ and $\Sigma_t = \text{diag} (\sigma_1, \cdots, \sigma_t) \in \mathbb{R}^{t \times t}$. Then

$$G = U \Sigma U^T = U_1 \Sigma_t U_1^T. \tag{12}$$

Denote $R = (\Sigma_t^2 + \lambda \Sigma_t)^{-\frac{1}{2}} \Sigma_t U_1^T L$ and let $R = M \Sigma_R N^T$ be the SVD of $R$, where $M$ and $N$ are orthogonal and $\Sigma_R$ is diagonal with rank$(\Sigma_R)$ = rank$(R)$ = $q$. Define the matrix $Z$ as $Z = U \text{diag} \left( (\Sigma_t^2 + \lambda \Sigma_t)^{-\frac{1}{2}} M, I_{n-t} \right)$, where diag$(A, B)$ is a block diagonal matrix. It follows that

$$Z^T \left( GLL^T G \right) Z = \begin{pmatrix} \tilde{\Sigma} & 0 \\ 0 & 0 \end{pmatrix}, \ Z^T \left( GG + \lambda G \right) Z = \begin{pmatrix} I_t & 0 \\ 0 & 0 \end{pmatrix}, \tag{13}$$

where $\tilde{\Sigma} = (\Sigma_R)^2$ is diagonal with non-increasing diagonal entries. It can be verified that

$$
\begin{aligned}
f(L, P) &\leq \text{trace} \left( \tilde{\Sigma} \right) = \text{trace} \left( (GG + \lambda G)^+ GLL^T G \right) \\
&= \text{trace} \left( L^T G (GG + \lambda G)^+ GL \right) \\
&= \text{trace} \left( L^T \left( I_n - (I_n + \frac{1}{\lambda} G)^{-1} \right) L \right),
\end{aligned} \tag{14}
$$

where the equality holds when $P = XH$ and $H$ consists of the first $q$ columns of $Z$. $\qquad \square$

### 3.1 Computing the Weighted Cluster Matrix $L$

The weighted cluster indicator matrix $L$ solving the maximization problem in Eq. (11) can be computed by solving a kernel K-means problem [5] with the kernel Gram matrix given by

$$\tilde{G} = I_n - \left(I_n + \frac{1}{\lambda}G\right)^{-1}. \tag{15}$$

Thus, DisCluster is equivalent to a kernel K-means problem. We call the algorithm Discriminative K-means (DisKmeans).

### 3.2 Constructing the Kernel Gram Matrix via Subspace Selection

The kernel Gram matrix in Eq. (15) can be expressed as

$$\tilde{G} = U \text{ diag}\left(\sigma_1/(\lambda + \sigma_1), \sigma_2/(\lambda + \sigma_2), \cdots, \sigma_n/(\lambda + \sigma_n)\right) U^T. \tag{16}$$

Recall that the original DisCluster algorithm involves alternating LDA subspace selection and clustering. The analysis above shows that the LDA subspace selection in DisCluster essentially constructs a kernel Gram matrix for clustering. More specifically, all the eigenvectors in $G$ is kept unchanged, while the following transformation is applied to the eigenvalues:

$$\Phi(\sigma) = \sigma/(\lambda + \sigma).$$

This elucidates the nature of the subspace selection procedure in DisCluster. The clustering algorithm is dramatically simplified by removing the iterative subspace selection. We thus address issues (1)–(3) in Section 1. The last issue will be addressed in Section 4 below.

### 3.3 Connection with Other Clustering Approaches

Consider the limiting case when $\lambda \to \infty$. It follows from Eq. (16) that $\tilde{G} \to G/\lambda$. The optimal $L$ is thus given by solving the following maximization problem:

$$\arg\max_L \text{trace}\left(L^T G L\right).$$

The solution is given by the standard K-means clustering [4, 5].

Consider the other extreme case when $\lambda \to 0$. It follows from Eq. (16) that $\tilde{G} \to U_1^T U_1$. Note that the columns of $U_1$ form the full set of (normalized) principal components [9]. Thus, the algorithm is equivalent to clustering in the (full) principal component space.

## 4 DisKmeans$_\lambda$: Discriminative K-means with Automatically Tuned $\lambda$

Our experiments show that the value of the regularization parameter $\lambda$ has a significant impact on the performance of DisKmeans. In this section, we show how to incorporate the automatic tuning of $\lambda$ into the optimization framework, thus addressing issue (4) in Section 1.

The maximization problem in Eq. (11) is equivalent to the minimization of the following function:

$$\text{trace}\left(L^T\left(I_n + \frac{1}{\lambda}G\right)^{-1}L\right). \tag{17}$$

It is clear that a small value of $\lambda$ leads to a small value of the objective function in Eq. (17). To overcome this problem, we include an additional penalty term to control the eigenvalues of the matrix $I_n + \frac{1}{\lambda}G$. This leads to the following optimization problem:

$$\min_{L,\lambda} g(L,\lambda) \equiv \text{trace}\left(L^T\left(I_n + \frac{1}{\lambda}G\right)^{-1}L\right) + \log\det\left(I_n + \frac{1}{\lambda}G\right). \tag{18}$$

Note that the objective function in Eq. (18) is closely related to the negative *log marginal likelihood* function in Gaussian Process [12] with $I_n + \frac{1}{\lambda}G$ as the covariance matrix. We have the following main result for this section:

**Theorem 4.1.** *Let $G$ be the Gram matrix defined above and let $L$ be a given weighted cluster indicator matrix. Let $G = U\Sigma U^T = U_1\Sigma_t U_1^T$ be the SVD of $G$ with $\Sigma_t = \text{diag}(\sigma_1, \cdots, \sigma_t)$ as in Eq. (12), and $a_i$ be the $i$-th diagonal entry of the matrix $U_1^T LL^T U_1$. Then for a fixed $L$,*

the optimal $\lambda^*$ solving the optimization problem in Eq. (18) is given by minimizing the following objective function:

$$\sum_{i=1}^{t} \frac{\lambda a_i}{\lambda + \sigma_i} + \log\left(1 + \frac{\sigma_i}{\lambda}\right). \tag{19}$$

*Proof.* Let $U = [U_1, U_2]$, that is, $U_2$ is the orthogonal complement of $U_1$. It follows that

$$\log \det\left(I_n + \frac{1}{\lambda}G\right) = \log \det\left(I_t + \frac{1}{\lambda}\Sigma_1\right) = \sum_{i=1}^{t} \log\left(1 + \sigma_i/\lambda\right). \tag{20}$$

$$\text{trace}\left(L^T \left(I_n + \frac{1}{\lambda}G\right)^{-1} L\right) = \text{trace}\left(L^T U_1 \left(I_t + \frac{1}{\lambda}\Sigma_t\right)^{-1} U_1^T L\right) + \text{trace}\left(L^T U_2 U_2^T L\right)$$

$$= \sum_{i=1}^{t} (1 + \sigma_i/\lambda)^{-1} a_i + \text{trace}\left(L^T U_2 U_2^T L\right), \tag{21}$$

The result follows as the second term in Eq. (21), $\text{trace}\left(L^T U_2 U_2^T L\right)$, is a constant. $\square$

We can thus solve the optimization problem in Eq. (18) iteratively as follows: For a fixed $\lambda$, we update $L$ by maximizing the objective function in Eq. (17), which is equivalent to the DisKmeans algorithm; for a fixed $L$, we update $\lambda$ by minimizing the objective function in Eq. (19), which is a single-variable optimization and can be solved efficiently using the line search method. We call the algorithm DisKmeans$_\lambda$, whose solution depends on the initial value of $\lambda$.

## 5    Kernel DisKmeans: Nonlinear Discriminative K-means using the kernels

The DisKmeans algorithm can be easily extended to deal with nonlinear data using the kernel trick. Kernel methods [14] work by mapping the data into a high-dimensional feature space $\mathcal{F}$ equipped with an inner product through a nonlinear mapping $\phi : \mathbb{R}^m \rightarrow \mathcal{F}$. The nonlinear mapping can be implicitly specified by a symmetric *kernel function* $K$, which computes the inner product of the images of each data pair in the feature space. For a given training data set $\{x_i\}_{i=1}^n$, the kernel Gram matrix $G_K$ is defined as follows: $G_K(i,j) = (\phi(x_i), \phi(x_j))$. For a given $G_K$, the weighted cluster matrix $L = [L_1, \cdots, L_k]$ in kernel DisKmeans is given by minimizing the following objective function:

$$\text{trace}\left(L^T \left(I_n + \frac{1}{\lambda}G_K\right)^{-1} L\right) = \sum_{j=1}^{k} L_j^T \left(I_n + \frac{1}{\lambda}G_K\right)^{-1} L_j. \tag{22}$$

The performance of kernel DisKmeans is dependent on the choice of the kernel Gram matrix. Following [11], we assume that $G_K$ is restricted to be a convex combination of a given set of kernel Gram matrices $\{G_i\}_{i=1}^\ell$ as $G_K = \sum_{i=1}^\ell \theta_i G_i$, where the coefficients $\{\theta_i\}_{i=1}^\ell$ satisfy $\sum_{i=1}^\ell \theta_i \text{trace}(G_i) = 1$ and $\theta_i \geq 0 \ \forall i$. If $L$ is given, the optimal coefficients $\{\theta_i\}_{i=1}^\ell$ may be computed by solving a Semidefinite programming (SDP) problem as follows:

**Theorem 5.1.** *Let $G_K$ be constrained to be a convex combination of a given set of kernel matrices $\{G_i\}_{i=1}^\ell$ as $G_K = \sum_{i=1}^\ell \theta_i G_i$ satisfying the constraints defined above. Then the optimal $G_K$ minimizing the objective function in Eq. (22) is given by solving the following SDP problem:*

$$\min_{t_1, \cdots, t_k, \theta} \sum_{j=1}^{k} t_j$$

$$s.t. \qquad \begin{pmatrix} I_n + \frac{1}{\lambda}\sum_{i=1}^{\ell}\theta_i \tilde{G}_i & L_j \\ L_j^T & t_j \end{pmatrix} \succeq 0, \ for \ j = 1, \cdots, k,$$

$$\theta_i \geq 0 \ \forall i, \quad \sum_{i=1}^{\ell} \theta_i \ trace(G_i) = 1. \tag{23}$$

*Proof.* It follows as $L_j^T \left(I_n + \frac{1}{\lambda}G_K\right)^{-1} L_j \leq t_i$ is equivalent to $\begin{pmatrix} I + \frac{1}{\lambda}\sum_{i=1}^{\ell}\theta_i \tilde{G}_i & L_j \\ L_j^T & t_j \end{pmatrix} \succeq 0.$ $\square$

This leads to an iterative algorithm alternating between the computation of the kernel Gram matrix $G_K$ and the computation of the cluster indicator matrix $L$. The parameter $\lambda$ can also be incorporated into the SDP formulation by treating the identity matrix $I_n$ as one of the kernel Gram matrix as in [11]. The algorithm is named Kernel DisKmeans$_\lambda$. Note that unlike the kernel learning in [11], the class label information is not available in our formulation.

## 6 Empirical Study

In this section, we empirically study the properties of DisKmeans and its variants, and evaluate the performance of the proposed algorithms in comparison with several other representative algorithms, including Locally Linear Embedding (LLE) [13] and Laplacian Eigenmap (Leigs) [1].

**Experiment Setup:** All algorithms were implemented using Matlab and experiments were conducted on a PENTIUM IV 2.4G PC with 1.5GB RAM. We test these algorithms on eight benchmark data sets. They are five UCI data sets [2]: banding, soybean, segment, satimage, pendigits; one biological data set: leukemia (`http://www.upo.es/eps/aguilar/datasets.html`) and two image data sets: ORL (`http://www.uk.research.att.com/facedatabase.html`, sub-sampled to a size of 100*100

Table 1: Summary of benchmark data sets

| Data Set | # DIM ($m$) | # INST ($n$) | # CL ($k$) |
|----------|------|------|------|
| banding | 29 | 238 | 2 |
| soybean | 35 | 562 | 15 |
| segment | 19 | 2309 | 7 |
| pendigits | 16 | 10992 | 10 |
| satimage | 36 | 6435 | 6 |
| leukemia | 7129 | 72 | 2 |
| ORL | 10304 | 100 | 10 |
| USPS | 256 | 9298 | 10 |

= 10000 from 10 persons) and USPS (`ftp://ftp.kyb.tuebingen.mpg.de/pub/bs/data/`). See Table 1 for more details. To make the results of different algorithms comparable, we first run $K$-means and the clustering result of $K$-means is used to construct the set of $k$ initial centroids, for all experiments. This process is repeated for 50 times with different sub-samples from the original data sets. We use two standard measurements: the accuracy (ACC) and the normalized mutual information (NMI) to measure the performance.

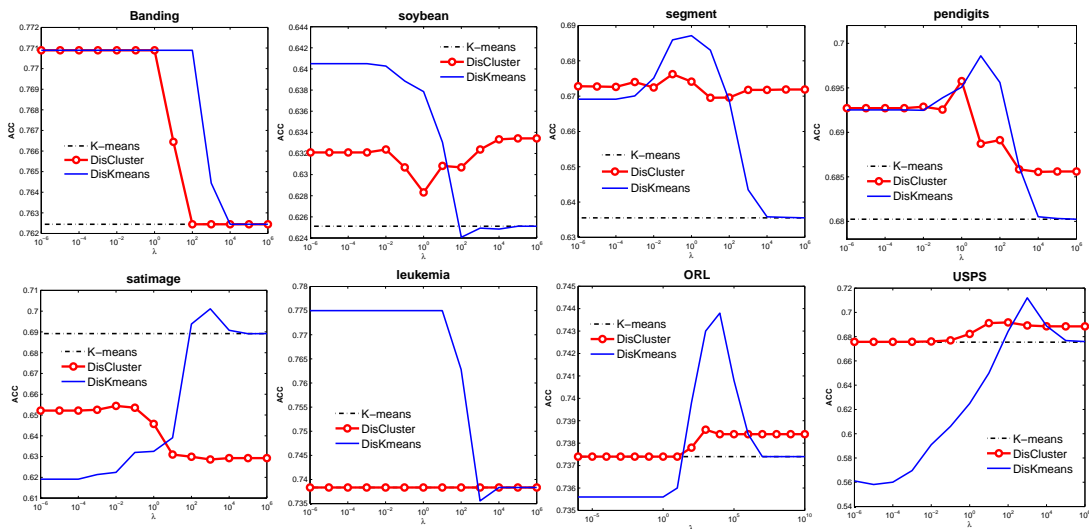

Figure 1: The effect of the regularization parameter $\lambda$ on DisKmeans and Discluster.

**Effect of the regularization parameter $\lambda$:** Figure 1 shows the accuracy ($y$-axis) of DisKmeans and DisCluster for different $\lambda$ values ($x$-axis). We can observe that $\lambda$ has a significant impact on the performance of DisKmeans. This justifies the development of an automatic parameter tuning process in Section 4. We can also observe from the figure that when $\lambda \to \infty$, the performance of DisKmeans approaches to that of $K$-means on all eight benchmark data sets. This is consistent with our theoretical analysis in Section 3.3. It is clear that in many cases, $\lambda = 0$ is not the best choice.

**Effect of parameter tuning in DisKmeans$_\lambda$:** Figure 2 shows the accuracy of DisKmeans$_\lambda$ using 4 data sets. In the figure, the $x$-axis denotes the different $\lambda$ values used as the starting point for DisKmeans$_\lambda$. The result of DisKmeans (without parameter tuning) is also presented for comparison. We can observe from the figure that in many cases the tuning process is able to significantly improve the performance. We observe similar trends on other four data sets and the results are omitted.

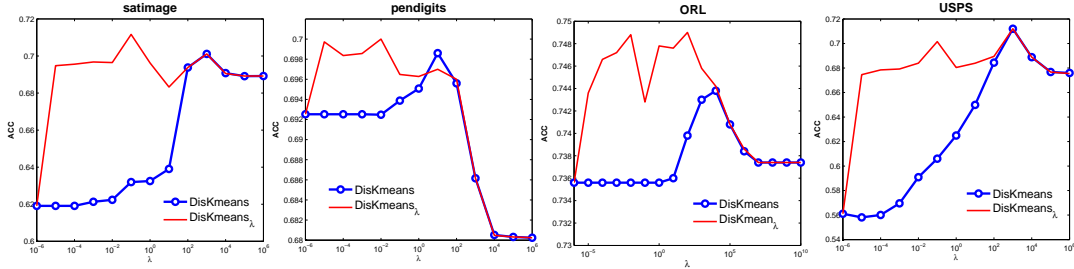

Figure 2: The effect of the parameter tuning in DisKmeans$_\lambda$ using 4 data sets. The $x$-axis denotes the different $\lambda$ values used as the starting point for DisKmeans$_\lambda$.

Figure 2 also shows that the tuning process is dependent on the initial value of $\lambda$ due to its non-convex optimization, and when $\lambda \to \infty$, the effect of the tuning process become less pronounced. Our results show that a value of $\lambda$, which is neither too large nor too small works well.

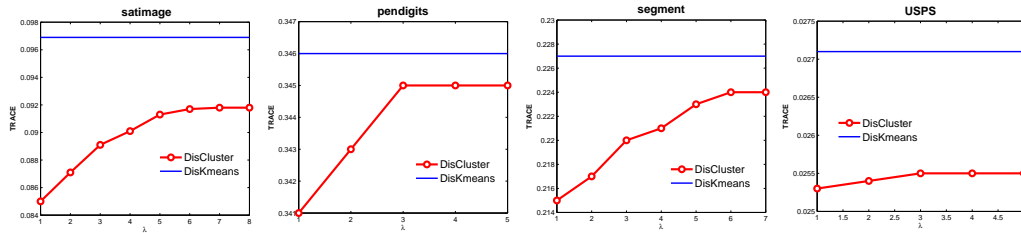

Figure 3: Comparison of the trace value achieved by DisKmean and DisCluster. The $x$-axis denotes the number of iterations in Discluster. The trace value of DisCluster is bounded from above by that of DisKmean.

**DisKmean versus DisCluster:** Figure 3 compares the trace value achieved by DisKmean and the trace value achieved in each iteration of DisCluster on 4 data sets for a fixed $\lambda$. It is clear that the trace value of DisCluster increases in each iteration but is bounded from above by that of DisKmean. We observe a similar trend on the other four data sets and the results are omitted. This is consistent with our analysis in Section 3 that both algorithms optimize the same objective function, and DisKmean is a direct approach for the trace maximization without the iterative process.

**Clustering evaluation:** Table 2 presents the accuracy (ACC) and normalized mutual information (NMI) results of various algorithms on all eight data sets. In the table, DisKmeans (or DisCluster) with "max" and "ave" stands for the maximal and average performance achieved by DisKmeans and DisCluster using $\lambda$ from a wide range between $10^{-6}$ and $10^6$. We can observe that DisKmeans$_\lambda$ is competitive with other algorithms. It is clear that the average performance of DisKmeans$_\lambda$ is robust against different initial values of $\lambda$. We can also observe that the average performance of DisKmeans and DisCluster is quite similar, while DisCluster is less sensitive to the value of $\lambda$.

## 7 Conclusion

In this paper, we analyze the discriminative clustering (DisCluster) framework, which integrates subspace selection and clustering. We show that the iterative subspace selection and clustering in DisCluster is equivalent to kernel K-means with a specific kernel Gram matrix. We then propose the DisKmeans algorithm for simultaneous LDA subspace selection and clustering, as well as an automatic parameter tuning procedure. The connection between DisKmeans and several other clustering algorithms is also studied. The presented analysis and algorithms are verified through experiments on a collection of benchmark data sets.

We present the nonlinear extension of DisKmeans in Section 5. Our preliminary studies have shown the effectiveness of Kernel DisKmeans$_\lambda$ in learning the kernel Gram matrix. However, the SDP formulation is limited to small-sized problems. We plan to explore efficient optimization techniques for this problem. Partial label information may be incorporated into the proposed formulations. This leads to semi-supervised clustering [3]. We plan to examine various semi-learning techniques within the proposed framework and their effectiveness for clustering from both labeled and unlabeled data.

Table 2: Accuracy (ACC) and Normalized Mutual Information (NMI) results on 8 data sets. "max" and "ave" stand for the maximal and average performance achieved by DisKmeans and DisCluster using $\lambda$ from a wide range of values between $10^{-6}$ and $10^6$. We present the result of DisKmeans$_\lambda$ with different initial $\lambda$ values. LLE stands for Local Linear Embedding and LEI for Laplacian Eigenmap. "AVE" stands for the mean of ACC or NMI on 8 data sets for each algorithm.

| Data Sets | DisKmeans | | DisCluster | | DisKmeans$_\lambda$ | | | | LLE | LEI |
|---|---|---|---|---|---|---|---|---|---|---|
| | max | ave | max | ave | $10^{-2}$ | $10^{-1}$ | $10^0$ | $10^1$ | | |
| ACC | | | | | | | | | | |
| banding | 0.771 | 0.768 | 0.771 | 0.767 | 0.771 | 0.771 | 0.771 | 0.771 | 0.648 | 0.764 |
| soybean | 0.641 | 0.634 | 0.633 | 0.632 | 0.639 | 0.639 | 0.638 | 0.637 | 0.630 | 0.649 |
| segment | 0.687 | 0.664 | 0.676 | 0.672 | 0.664 | 0.659 | 0.671 | 0.680 | 0.594 | 0.663 |
| pendigits | 0.699 | 0.690 | 0.696 | 0.690 | 0.700 | 0.696 | 0.696 | 0.697 | 0.599 | 0.697 |
| satimage | 0.701 | 0.651 | 0.654 | 0.642 | 0.696 | 0.712 | 0.696 | 0.683 | 0.627 | 0.663 |
| leukemia | 0.775 | 0.763 | 0.738 | 0.738 | 0.738 | 0.753 | 0.738 | 0.738 | 0.714 | 0.686 |
| ORL | 0.744 | 0.738 | 0.739 | 0.738 | 0.749 | 0.743 | 0.748 | 0.748 | 0.733 | 0.317 |
| USPS | 0.712 | 0.628 | 0.692 | 0.683 | 0.684 | 0.702 | 0.680 | 0.684 | 0.631 | 0.700 |
| AVE | 0.716 | 0.692 | 0.700 | 0.695 | 0.705 | 0.709 | 0.705 | 0.705 | 0.647 | 0.642 |
| NMI | | | | | | | | | | |
| banding | 0.225 | 0.221 | 0.225 | 0.219 | 0.225 | 0.225 | 0.225 | 0.225 | 0.093 | 0.213 |
| soybean | 0.707 | 0.701 | 0.698 | 0.696 | 0.706 | 0.707 | 0.704 | 0.704 | 0.691 | 0.709 |
| segment | 0.632 | 0.612 | 0.615 | 0.608 | 0.629 | 0.625 | 0.628 | 0.632 | 0.539 | 0.618 |
| pendigits | 0.669 | 0.656 | 0.660 | 0.654 | 0.661 | 0.658 | 0.658 | 0.660 | 0.577 | 0.645 |
| satimage | 0.593 | 0.537 | 0.551 | 0.541 | 0.597 | 0.608 | 0.596 | 0.586 | 0.493 | 0.548 |
| leukemia | 0.218 | 0.199 | 0.163 | 0.163 | 0.163 | 0.185 | 0.163 | 0.163 | 0.140 | 0.043 |
| ORL | 0.794 | 0.789 | 0.789 | 0.788 | 0.800 | 0.795 | 0.801 | 0.800 | 0.784 | 0.327 |
| USPS | 0.647 | 0.544 | 0.629 | 0.613 | 0.612 | 0.637 | 0.609 | 0.612 | 0.569 | 0.640 |
| AVE | 0.561 | 0.532 | 0.541 | 0.535 | 0.549 | 0.555 | 0.548 | 0.548 | 0.486 | 0.468 |

# Acknowledgments

This research is sponsored by the National Science Foundation Grant IIS-0612069.

# References

[1] M. Belkin and P. Niyogi. Laplacian eigenmaps for dimensionality reduction and data representation. In *NIPS*, 2003.

[2] C.L. Blake and C.J. Merz. UCI repository of machine learning databases, 1998.

[3] O. Chapelle, B. Schölkopf, and A. Zien. *Semi-Supervised Learning*. The MIT Press, 2006.

[4] I. S. Dhillon, Y. Guan, and B. Kulis. A unified view of kernel k-means, spectral clustering and graph partitioning. Technical report, Department of Computer Sciences, University of Texas at Austin, 2005.

[5] C. Ding and T. Li. Adaptive dimension reduction using discriminant analysis and k-means clustering. In *ICML*, 2007.

[6] J. H. Friedman. Regularized discriminant analysis. *JASA*, 84(405):165–175, 1989.

[7] K. Fukunaga. *Introduction to Statistical Pattern Classification*. Academic Press.

[8] G. H. Golub and C. F. Van Loan. *Matrix Computations*. The Johns Hopkins Univ. Press, 1996.

[9] I.T. Jolliffe. *Principal Component Analysis*. Springer; 2nd edition, 2002.

[10] F. De la Torre Frade and T. Kanade. Discriminative cluster analysis. In *ICML*, pages 241–248, 2006.

[11] G.R.G. Lanckriet, N. Cristianini, P. Bartlett, L. E. Ghaoui, and M. I. Jordan. Learning the kernel matrix with semidefinite programming. *JMLR*, 5:27–72, 2004.

[12] C.E. Rasmussen and C.K.I. Williams. *Gaussian Processes for Machine Learning*. The MIT Press, 2006.

[13] S. T. Roweis and L. K. Saul. Nonlinear dimensionality reduction by locally linear embedding. *Science*, 290:2323–2326, 2000.

[14] B. Schölkopf and A. Smola. *Learning with Kernels: Support Vector Machines, Regularization, Optimization and Beyond*. MIT Press, 2002.

[15] L. Vandenberghe and S. Boyd. Semidefinite programming. *SIAM Review*, 38:49–95, 1996.

[16] J. Ye, Z. Zhao, and H. Liu. Adaptive distance metric learning for clustering. In *CVPR*, 2007.

